# Speech Recognition using SVMs

**Nathan Smith**
Cambridge University
Engineering Dept
Cambridge, CB2 1PZ, U.K.
*nds1002@eng.cam.ac.uk*

**Mark Gales**
Cambridge University
Engineering Dept
Cambridge, CB2 1PZ, U.K.
*mjfg@eng.cam.ac.uk*

## Abstract

An important issue in applying SVMs to speech recognition is the ability to classify variable length sequences. This paper presents extensions to a standard scheme for handling this variable length data, the Fisher score. A more useful mapping is introduced based on the likelihood-ratio. The score-space defined by this mapping avoids some limitations of the Fisher score. Class-conditional generative models are directly incorporated into the definition of the score-space. The mapping, and appropriate normalisation schemes, are evaluated on a speaker-independent isolated letter task where the new mapping outperforms both the Fisher score and HMMs trained to maximise likelihood.

## 1 Introduction

Speech recognition is a complex, dynamic classification task. State-of-the-art systems use Hidden Markov Models (HMMs), either trained to maximise likelihood or discriminatively, to achieve good levels of performance. One of the reasons for the popularity of HMMs is that they readily handle the variable length data sequences which result from variations in word sequence, speaker rate and accent. Support Vector Machines (SVMs) [1] are a powerful, discriminatively-trained technique that have been shown to work well on a variety of tasks. However they are typically only applied to static binary classification tasks. This paper examines the application of SVMs to speech recognition. There are two major problems to address. First, how to handle the variable length sequences. Second, how to handle multi-class decisions. This paper only concentrates on dealing with variable length sequences. It develops our earlier research in [2] and is detailed more fully in [7]. A similar approach for protein classification is adopted in [3].

There have been a variety of methods proposed to map variable length sequences to vectors of fixed dimension. These include vector averaging and selecting a 'representative' number of observations from each utterance. However, these methods may discard useful information. This paper adopts an approach similar to that of [4] which makes use of all the available data. Their scheme uses generative probability models of the data to define a mapping into a fixed dimension space, the Fisher score-space. When incorporated within an SVM kernel, the kernel is known as the Fisher kernel. Relevant regularisation issues are discussed in [5]. This paper

examines the suitability of the Fisher kernel for classification in speech recognition and proposes an alternative, more useful, kernel. In addition some normalisation issues associated with using this kernel for speech recognition are addressed.

Initially a general framework for defining alternative score-spaces is required. First, define an observation sequence as $\boldsymbol{O} = (\boldsymbol{o}_1, \ldots \boldsymbol{o}_t, \ldots \boldsymbol{o}_T)$ where $\boldsymbol{o}_t \in \mathbb{R}^D$, and a set of generative probability models of the observation sequences as $\mathcal{P} = \{p_k(\boldsymbol{O}|\boldsymbol{\theta}_k)\}$, where $\boldsymbol{\theta}_k$ is the vector of parameters for the $k$th member of the set. The observation sequence $\boldsymbol{O}$ can be mapped into a vector of fixed dimension [4],

$$\varphi^f_{\hat{F}}(\boldsymbol{O}) \quad = \quad \varphi_{\hat{F}} f\big(\{p_k(\boldsymbol{O}|\boldsymbol{\theta}_k)\}\big) \tag{1}$$

$f(\cdot)$ is the *score-argument* and is a function of the members of the set of generative models $\mathcal{P}$. $\varphi_{\hat{F}}$ is the *score-mapping* and is defined using a *score-operator* $\hat{F}$. $\varphi^f_{\hat{F}}(\boldsymbol{O})$ is the *score* and occupies the fixed-dimension *score-space*. Our investigation of score-spaces falls into three divisions. What are the best generative models, score-arguments and score-operators to use?

## 2 Score-spaces

As HMMs have proved successful in speech recognition, they are a natural choice as the generative models for this task. In particular HMMs with state output distributions formed by Gaussian mixture models. There is also the choice of the score-argument. For a two-class problem, let $p_i(\boldsymbol{O}|\boldsymbol{\theta}_i)$ represent a generative model, where $i = \{g, 1, 2\}$ ($g$ denotes the global 2-class generative model, and 1 and 2 denote the class-conditional generative models for the two competing classes). Previous schemes have used the log of a single generative model, $\ln p_i(\boldsymbol{O}|\boldsymbol{\theta}_i)$ representing either both classes as in the original Fisher score ($i = g$) [4], or one of the classes ($i = 1$ or 2) [6]. This score-space is termed the *likelihood score-space*, $\varphi^{\mathrm{lik}}_{\hat{F}}(\boldsymbol{O})$. The score-space proposed in this paper uses the log of the ratio of the two class-conditional generative models, $\ln(p_1(\boldsymbol{O}|\boldsymbol{\theta}_1)/p_2(\boldsymbol{O}|\boldsymbol{\theta}_2))$ where $\boldsymbol{\theta} = [\boldsymbol{\theta}_1^\top, \boldsymbol{\theta}_2^\top]^\top$. The corresponding score-space is called the *likelihood-ratio score-space*, $\varphi^{\mathrm{lr}}_{\hat{F}}(\boldsymbol{O})$. Thus,

$$\varphi^{\mathrm{lik}}_{\hat{F}}(\boldsymbol{O}) \quad = \quad \varphi_{\hat{F}} \ln p_i(\boldsymbol{O}|\boldsymbol{\theta}_i) \tag{2}$$

$$\varphi^{\mathrm{lr}}_{\hat{F}}(\boldsymbol{O}) \quad = \quad \varphi_{\hat{F}} \ln\left[\frac{p_1(\boldsymbol{O}|\boldsymbol{\theta}_1)}{p_2(\boldsymbol{O}|\boldsymbol{\theta}_2)}\right] \tag{3}$$

The likelihood-ratio score-space can be shown to avoid some of the limitations of the likelihood score-space, and may be viewed as a generalisation of the standard generative model classifier. These issues will be discussed later.

Having proposed forms for the generative models and score-arguments, the score-operators must be selected. The original score-operator in [4] was the 1st-order derivative operator applied to HMMs with discrete output distributions. Consider a continuous density HMM with N emitting states, $j \in \{1 \ldots N\}$. Each state, $j$, has an output distribution formed by a mixture of $K$ Gaussian components, $\mathcal{N}(\boldsymbol{\mu}_{jk}, \Sigma_{jk})$ where $k \in \{1 \ldots K\}$. Each component has parameters of weight $w_{jk}$, mean $\boldsymbol{\mu}_{jk}$ and covariance $\Sigma_{jk}$. The 1st-order derivatives of the log probability of the sequence $\boldsymbol{O}$ with respect to the model parameters are given below[1], where the derivative operator has been defined to give column vectors,

$$\nabla_{\boldsymbol{\mu}_{jk}} \ln p(\boldsymbol{O}|\boldsymbol{\theta}) \quad = \quad \sum_{t=1}^{T} \gamma_{jk}(t) S^\top_{[t,jk]} \tag{4}$$

$$\nabla_{\text{vec}(\Sigma_{jk})} \ln p(\boldsymbol{O}|\boldsymbol{\theta}) = \sum_{t=1}^{T} \gamma_{jk}(t) \frac{1}{2} \left[ -[\text{vec}(\Sigma_{jk}^{-1})]^{\top} + \{S_{[t,jk]} \otimes S_{[t,jk]}\} \right]^{\top} \quad (5)$$

$$\nabla_{w_{jk}} \ln p(\boldsymbol{O}|\boldsymbol{\theta}) = \sum_{t=1}^{T} \gamma_{jk}(t) \left[ \frac{1}{w_{jk}} - \frac{\gamma_{j1}(t)}{w_{j1}\gamma_{jk}(t)} \right] \quad (6)$$

$$\text{where} \qquad S_{[t,jk]} = (\boldsymbol{o}_t - \boldsymbol{\mu}_{jk})^{\top} \Sigma_{jk}^{-1} \quad (7)$$

$\gamma_{jk}(t)$ is the posterior probability of component $k$ of state $j$ at time $t$. Assuming the HMM is left-to-right with no skips and assuming that a state only appears once in the HMM (i.e. there is no state-tying), then the 1st-order derivative for the self-transition probability for state $j$, $a_{jj}$, is,

$$\nabla_{a_{jj}} \ln p(\boldsymbol{O}|\boldsymbol{\theta}) = \sum_{t=1}^{T} \left[ \frac{\gamma_j(t)}{a_{jj}} - \frac{1}{T a_{jj}(1-a_{jj})} \right] \quad (8)$$

The 1st-order derivatives for each Gaussian parameter and self-transition probability in the HMM can be spliced together into a 'super-vector' which is the *score*[2].

From the definitions above, the score for an utterance is a weighted sum of scores for individual observations. If the scores for the same utterance spoken at different speaking rates were calculated, they would lie in different regions of score-space simply because of differing numbers of observations. To ease the task of the classifier in score-space, the score-space may be normalised by the number of observations, called *sequence length normalisation*. Duration information can be retained in the derivatives of the transition probabilities. One method of normalisation redefines score-spaces using generative models trained to maximise a modified log likelihood function, $l_n(\boldsymbol{O}|\boldsymbol{\theta})$. Consider that state $j$ has entry time $\tau_j$ and duration $d_j$ (both in numbers of observations) and output probability $b_j(\boldsymbol{o}_t)$ for observation $\boldsymbol{o}_t$ [7]. So,

$$l_n(\boldsymbol{O}|\boldsymbol{\theta}) = \sum_{j=1}^{N} \frac{1}{d_j} \left( (d_j - 1) \ln a_{jj} + \ln a_{j(j+1)} + \sum_{t=\tau_j}^{\tau_j+d_j-1} (\ln b_j(\boldsymbol{o}_t)) \right) \quad (9)$$

It is not possible to maximise $l_n(\boldsymbol{O}|\boldsymbol{\theta})$ using the EM algorithm. Hill-climbing techniques could be used. However, in this paper, a simpler normalisation method is employed. The generative models are trained to maximise the standard likelihood function. Rather than define the score-space using standard state posteriors $\gamma_j(t)$ (the posterior probability of state $j$ at time $t$), it is defined on state posteriors normalised by the total state occupancy over the utterance. The standard component posteriors $\gamma_{jk}(t)$ are replaced in Equations 4 to 6 and 8 by their normalised form $\hat{\gamma}_{jk}(t)$,

$$\hat{\gamma}_{jk}(t) = \frac{\gamma_j(t)}{\sum_{\tau=1}^{T} \gamma_j(\tau)} \left( \frac{w_{jk} \mathcal{N}(\boldsymbol{o}_t; \boldsymbol{\mu}_{jk}, \Sigma_{jk})}{\sum_{i=1}^{K} w_{ji} \mathcal{N}(\boldsymbol{o}_t; \boldsymbol{\mu}_{ji}, \Sigma_{ji})} \right) \quad (10)$$

In effect, each derivative is divided by the sum of state posteriors. This is preferred to division by the total number of observations $T$ which assumes that when the utterance length varies, the occupation of every state in the state sequence is scaled by the same ratio. This is not necessarily the case for speech.

The nature of the score-space affects the discriminative power of classifiers built in the score-space. For example, the likelihood score-space defined on a two-class

generative model is susceptible to *wrap-around* [7]. This occurs when two different locations in acoustic-space map to a single point in score-subspace. As an example, consider two classes modelled by two widely-spaced Gaussians. If an observation is generated at the peak of the first Gaussian, then the derivative relative to the mean of that Gaussian is zero because $S_{[t,jk]}$ is zero (see Equation 4). However, the derivative relative to the mean of the distant second Gaussian is also zero because of a zero component posterior $\gamma_{jk}(t)$. A similar problem occurs with an observation generated at the peak of the second Gaussian. This ambiguity in mapping two possible locations in acoustic-space to the zero of the score-subspace of the means represents a wrapping of the acoustic space onto this subspace. This also occurs in the subspace of the variances. Thus wrap-around can increase class confusion. A likelihood-ratio score-space defined on these two Gaussians does not suffer wrap-around since the component posteriors for each Gaussian are forced to unity.

So far, only 1st-order derivative score-operators have been considered. It is possible to include the zeroth, 2nd and higher-order derivatives. Of course there is an interaction between the score-operator and the score-argument. For example, the zeroth-order derivative for the likelihood score-space is expected to be less useful than its counter-part in the likelihood-ratio score-space because of its greater sensitivity to acoustic conditions. A principled approach to using derivatives in score-spaces would be useful. Consider the simple case of true class-conditional generative models $p_1(\boldsymbol{O}|\boldsymbol{\theta}_1)$ and $p_2(\boldsymbol{O}|\boldsymbol{\theta}_2)$ with respective estimates of the same functional form $p_1(\boldsymbol{O}|\hat{\boldsymbol{\theta}}_1)$ and $p_2(\boldsymbol{O}|\hat{\boldsymbol{\theta}}_2)$. Expressing the true models as Taylor series expansions about the parameter estimates $\hat{\boldsymbol{\theta}}_1$ and $\hat{\boldsymbol{\theta}}_2$ (see [7] for more details, and [3]),

$$
\begin{aligned}
\ln p_i(\boldsymbol{O}|\boldsymbol{\theta}_i) &= \ln p_i(\boldsymbol{O}|\hat{\boldsymbol{\theta}}_i) + (\boldsymbol{\theta}_i - \hat{\boldsymbol{\theta}}_i)^\top \nabla_{\hat{\boldsymbol{\theta}}_i} \ln p_i(\boldsymbol{O}|\hat{\boldsymbol{\theta}}_i) \\
&\quad + \frac{1}{2}(\boldsymbol{\theta}_i - \hat{\boldsymbol{\theta}}_i)^\top [\nabla_{\hat{\boldsymbol{\theta}}_i} \nabla_{\hat{\boldsymbol{\theta}}_i}^\top \ln p_i(\boldsymbol{O}|\hat{\boldsymbol{\theta}}_i)](\boldsymbol{\theta}_i - \hat{\boldsymbol{\theta}}_i) + O\big(\boldsymbol{\theta}_i(\cdot)^3\big) \\
&= \boldsymbol{w}_i^\top [1, \nabla_{\hat{\boldsymbol{\theta}}_i}^\top, \mathrm{vec}(\nabla_{\hat{\boldsymbol{\theta}}_i} \nabla_{\hat{\boldsymbol{\theta}}_i}^\top)^\top \ldots]^\top \ln p_i(\boldsymbol{O}|\hat{\boldsymbol{\theta}}_i) \quad (11)
\end{aligned}
$$

The output from the operator in square brackets is an infinite number of derivatives arranged as a column vector. $\boldsymbol{w}_i$ is also a column vector. The expressions for the two true models can be incorporated into an optimal minimum Bayes error decision rule as follows, where $\hat{\boldsymbol{\theta}} = [\hat{\boldsymbol{\theta}}_1^\top, \hat{\boldsymbol{\theta}}_2^\top]^\top$, $\boldsymbol{w} = [\boldsymbol{w}_1^\top, \boldsymbol{w}_2^\top]^\top$, and $b$ encodes the class priors,

$$
\begin{aligned}
\ln p_1(\boldsymbol{O}|\boldsymbol{\theta}_1) - \ln p_2(\boldsymbol{O}|\boldsymbol{\theta}_2) + b &= 0 \\
\boldsymbol{w}_1^\top [1, \nabla_{\hat{\boldsymbol{\theta}}_1}^\top, \mathrm{vec}(\nabla_{\hat{\boldsymbol{\theta}}_1} \nabla_{\hat{\boldsymbol{\theta}}_1}^\top)^\top \ldots]^\top \ln p_1(\boldsymbol{O}|\hat{\boldsymbol{\theta}}_1) - \\
\boldsymbol{w}_2^\top [1, \nabla_{\hat{\boldsymbol{\theta}}_2}^\top, \mathrm{vec}(\nabla_{\hat{\boldsymbol{\theta}}_2} \nabla_{\hat{\boldsymbol{\theta}}_2}^\top)^\top \ldots]^\top \ln p_2(\boldsymbol{O}|\hat{\boldsymbol{\theta}}_2) + b &= 0 \\
\boldsymbol{w}^\top [1, \nabla_{\hat{\boldsymbol{\theta}}}^\top, \mathrm{vec}(\nabla_{\hat{\boldsymbol{\theta}}} \nabla_{\hat{\boldsymbol{\theta}}}^\top)^\top \ldots]^\top \ln \frac{p_1(\boldsymbol{O}|\hat{\boldsymbol{\theta}}_1)}{p_2(\boldsymbol{O}|\hat{\boldsymbol{\theta}}_2)} + b &= 0 \\
\boldsymbol{w}^\top \varphi^{\mathrm{lr}}(\boldsymbol{O}) + b &= 0 \quad (12)
\end{aligned}
$$

$\varphi^{\mathrm{lr}}(\boldsymbol{O})$ is a score in the likelihood-ratio score-space formed by an infinite number of derivatives with respect to the parameter estimates $\hat{\boldsymbol{\theta}}$. Therefore, the optimal decision rule can be recovered by constructing a well-trained linear classifier in $\varphi^{\mathrm{lr}}(\boldsymbol{O})$. In this case, the standard *SVM margin* can be interpreted as the *log posterior margin*. This justifies the use of the likelihood-ratio score-space and encourages the use of higher-order derivatives. However, most HMMs used in speech recognition are 1st-order Markov processes but speech is a high-order or infinite-order Markov

process. Therefore, a linear decision boundary in the likelihood-ratio score-space defined on 1st-order Markov model estimates is unlikely to be sufficient for recovering the optimal decision rule due to model incorrectness. However, powerful non-linear classifiers may be trained in such a likelihood-ratio score-space to try to compensate for model incorrectness and approximate the optimal decision rule. SVMs with non-linear kernels such as polynomials or Gaussian Radial Basis Functions (GRBFs) may be used. Although gains are expected from incorporating higher-order derivatives into the score-space, the size of the score-space dramatically increases. Therefore, practical systems may truncate the likelihood-ratio score-space after the 1st-order derivatives, and hence use linear approximations to the Taylor series expansions[3]. However, an example of a 2nd-order derivative is $\nabla_{\boldsymbol{\mu}_{jk}}\left(\nabla_{\boldsymbol{\mu}_{jk}}^{\top} \ln p(\boldsymbol{O}|\boldsymbol{\theta})\right)$,

$$\nabla_{\boldsymbol{\mu}_{jk}}\left(\nabla_{\boldsymbol{\mu}_{jk}}^{\top} \ln p(\boldsymbol{O}|\boldsymbol{\theta})\right) \approx -\sum_{t=1}^{T} \gamma_{jk}(t)\Sigma_{jk}^{-1} \tag{13}$$

For simplicity the component posterior $\gamma_{jk}(t)$ is assumed independent of $\boldsymbol{\mu}_{jk}$. Once the score-space has been defined, an SVM classifier can be built in the score-space. If standard linear, polynomial or GRBF kernels are used in the score-space, then the space is assumed to have a Euclidean metric tensor. Therefore, the score-space should first be whitened (i.e. decorrelated and scaled) before the standard kernels are applied. Failure to perform such *score-space normalisation* for a linear kernel in score-space results in a kernel similar to the Plain kernel [5]. This is expected to perform poorly when different dimensions of score-space have different dynamic ranges [2]. Simple scaling has been found to be a reasonable approximation to full whitening and avoids inverting large matrices in [2] (though for classification of single observations rather than sequences, on a different database). The Fisher kernel in [4] uses the Fisher Information matrix to normalise the score-space. This is only an acceptable normalisation for a likelihood score-space under conditions that give a zero expectation in score-space. The appropriate SVM kernel to use between two utterances $\boldsymbol{O}_i$ and $\boldsymbol{O}_j$ in the normalised score-space is therefore the *Normalised kernel*, $k_{\mathrm{N}}(\boldsymbol{O}_i, \boldsymbol{O}_j)$ (where $\Sigma_{\mathrm{sc}}$ is the covariance matrix in score-space),

$$k_{\mathrm{N}}(\boldsymbol{O}_i, \boldsymbol{O}_j) = \varphi_{\hat{F}}^{f}(\boldsymbol{O}_i)^{\top}\Sigma_{\mathrm{sc}}^{-1}\varphi_{\hat{F}}^{f}(\boldsymbol{O}_j) \tag{14}$$

## 3 Experimental Results

The ISOLET speaker-independent isolated letter database [8] was used for evaluation. The data was coded at a 10 msec frame rate with a 25.6 msec window-size. The data was parameterised into 39-dimensional feature vectors including 12 MFCCs and a log energy term with corresponding delta and acceleration parameters. 240 utterances per letter from `isolet{1,2,3,4}` were used for training and 60 utterances per letter from `isolet5` for testing. There was no overlap between the training and test speakers. Two sets of letters were tested, the highly confusible E-set, {B C D E G P T V Z}, and the full 26 letters. The baseline HMM system was well-trained to maximise likelihood. Each letter was modelled by a 10-emitting state left-to-right continuous density HMM with no skips, and silence by a single emitting-state HMM with no skips. Each state output distribution had the same number of Gaussian components with diagonal covariance matrices. The models were tested using a Viterbi recogniser constrained to a silence-letter-silence network.

The baseline HMMs were used as generative models for SVM kernels. A modified version of $SVM^{light}$ Version 3.02 [9] was used to train 1v1 SVM classifiers on each possible class pairing. The sequence length normalisation in Equation 10, and simple scaling for score-space normalisation, were used during training and testing. Linear kernels were used in the normalised score-space, since they gave better performance than GRBFs of variable width and polynomial kernels of degree 2 (including homogeneous, inhomogeneous, and inhomogeneous with zero-mean score-space). The linear kernel did not require parameter-tuning and, in initial experiments, was found to be fairly insensitive to variations in the SVM trade-off parameter $C$. $C$ was fixed at 100, and biased hyperplanes were permitted. A variety of score-subspaces were examined. The abbreviations m, v, w and t refer to the score-subspaces $\nabla_{\boldsymbol{\mu}_{jk}} \ln p_i(\boldsymbol{O}|\boldsymbol{\theta}_i)$, $\nabla_{\text{vec}(\Sigma_{jk})} \ln p_i(\boldsymbol{O}|\boldsymbol{\theta}_i)$, $\nabla_{w_{jk}} \ln p_i(\boldsymbol{O}|\boldsymbol{\theta}_i)$ and $\nabla_{a_{jj}} \ln p_i(\boldsymbol{O}|\boldsymbol{\theta}_i)$ respectively. l refers to the log likelihood $\ln p_i(\boldsymbol{O}|\boldsymbol{\theta}_i)$ and r to the log likelihood-ratio $\ln[p_2(\boldsymbol{O}|\boldsymbol{\theta}_2)/p_1(\boldsymbol{O}|\boldsymbol{\theta}_1)]$. The binary SVM classification results (and, as a baseline, the binary HMM results) were combined to obtain a single classification for each utterance. This was done using a simple majority voting scheme among the full set of 1v1 binary classifiers (for tied letters, the relevant 1v1 classifiers were inspected and then, if necessary, random selection performed [2]).

Table 1: Error-rates for HMM baselines and SVM score-spaces (E-set)

| Num comp. per class per state | HMM | | SVM score-space | | |
|---|---|---|---|---|---|
| | min. Bayes error | majority voting | lik-ratio (stat. sign.) | lik (1-class) | lik (2-class) |
| 1 | 11.3 | 11.3 | 6.9 (99.8%) | 7.6 | 6.1 |
| 2 | 8.7 | 8.7 | 5.0 (98.9%) | 6.3 | 9.3 |
| 4 | 6.7 | 6.7 | 5.7 (13.6%) | 8.0 | 23.2 |
| 6 | 7.2 | 7.2 | 6.1 (59.5%) | 7.8 | 30.6 |

Table 1 compares the baseline HMM and SVM classifiers as the complexity of the generative models was varied. Statistical significance confidence levels are given in brackets comparing the baseline HMM and SVM classifiers with the same generative models, where 95% was taken as a significant result (confidence levels were defined by $(100 - P)$, where $P$ was given by McNemar's Test and was the percentage probability that the two classifiers had the same error rates and differences were simply due to random error; for this, a decision by random selection for tied letters was assigned to an 'undecided' class [7]). The baseline HMMs were comparable to reported results on the E-set for different databases [10]. The majority voting scheme gave the same performance as the minimum Bayes error scheme, indicating that majority voting was an acceptable multi-class scheme for the E-set experiments. For the SVMs, each likelihood-ratio score-space was defined using its competing class-conditional generative models and projected into a mr score-space. Each likelihood (1-class) score-space was defined using only the generative model for the first of its two classes, and projected into a ml score-space. Each likelihood (2-class) score-space was defined using a generative model for both of its classes, and projected into a ml score-space (the original Fisher score, which is a projection into its m score-subspace, was also tested but was found to yield slightly higher error rates). SVMs built using the likelihood-ratio score-space achieved lower error rates than HMM systems, as low as 5.0%. The likelihood (1-class) score-space performed slightly worse than the likelihood-ratio score-space because it contained about half the information and did not contain the log likelihood-ratio. In both cases, the optimum number of components in the generative models was 2 per state, possibly reflecting the gender division within each class. The likelihood (2-class) score-space performed poorly possibly because of wrap-around. However, there was an excep-

tion for generative models with 1 component per class per state (in total the models had 2 components per state since they modelled both classes). The 2 components per state did not generally reflect the gender division in the 2-class data, as first supposed, but the class division. A possible explanation is that each Gaussian component modelled a class with bi-modal distribution caused by gender differences. Most of the data modelled did not sit at the peaks of the two Gaussians and was not mapped to the ambiguous zero in score-subspace. Hence there was still sufficient class discrimination in score-space [7]. This task was too small to fully assess possible decorrelation in error structure between HMM and SVM classifiers [6].

Without scaling for score-space normalisation, the error-rate for the likelihood-ratio score-space defined on models with 2 components per state increased from 5.0% to 11.1%. Some likelihood-ratio mr score-spaces were then augmented with 2nd-order derivatives $\nabla_{\boldsymbol{\mu}_{jk}}(\nabla_{\boldsymbol{\mu}_{jk}}^{\top} \ln p(\boldsymbol{O}|\boldsymbol{\theta}))$. The resulting classifiers showed increases in error rate. The disappointing performance was probably due to the simplicity of the task, the independence assumption between component posteriors and component means, and the effect of noise with so few training scores in such large score-spaces.

It is known that some dimensions of feature-space are noisy and degrade classification performance. For this reason, experiments were performed which selected subsets of the likelihood-ratio score-space and then built SVM classifiers in those score-subspaces. First, the score-subspaces were selected by parameter type. Error rates for the resulting classifiers, otherwise identical to the baseline SVMs, are detailed in Table 2. Again, the generative models were class-conditional HMMs with 2 components per state. The log likelihood-ratio was shown to be a powerful discriminating feature[4]. Increasing the number of dimensions in score-space allowed more discriminative classifiers. There was more discrimination, or less noise, in the derivatives of the component means than the component variances. As expected in a dynamic task, the derivatives of the transitions were also useful since they contained some duration information.

Table 2: Error rates for subspaces of the likelihood-ratio score-space (E-set)

| score-space | error rate, % | score-space dimensionality |
|---|---|---|
| r | 8.5 | 1 |
| v | 7.2 | 1560 |
| m | 5.2 | 1560 |
| mv | 5.0 | 3120 |
| mvt | 4.4 | 3140 |
| wmvtr | 4.1 | 3161 |

Next, subsets of the mr and wmvtr score-spaces were selected according to dimensions with highest Fisher-ratios [7]. The lowest error rates for the mr and wmvtr score-spaces were respectively 3.7% at 200 dimensions and 3.2% at 500 dimensions (respectively significant at 99.1% and 99.7% confidence levels relative to the best HMM system with 4 components per state). Generally, adding the most discriminative dimensions lowered error-rate until less discriminative dimensions were added. For most binary classifiers, the most discriminative dimension was the log likelihood-ratio. As expected for the E-set, the most discriminative dimensions were dependent on initial HMM states. The low-order MFCCs and log energy term were the most important coefficients. Static, delta and acceleration streams were all useful.

The HMM and SVM classifiers were run on the full alphabet. The best HMM classifier, with 4 components per state, gave 3.4% error rate. Computational expense precluded a full optimisation of the SVM classifier. However, generative models with 2 components per state and a `wmvtr` score-space pruned to 500 dimensions by Fisher-ratios, gave a lower error rate of 2.1% (significant at a 99.0% confidence level relative to the HMM system). Preliminary experiments evaluating sequence length normalisation on the full alphabet and E-set are detailed in [7].

## 4  Conclusions

In this work, SVMs have been successfully applied to the classification of speech data. The paper has concentrated on the nature of the score-space when handling variable length speech sequences. The standard likelihood score-space of the Fisher kernel has been extended to the likelihood-ratio score-space, and normalisation schemes introduced. The new score-space avoids some of the limitations of the Fisher score-space, and incorporates the class-conditional generative models directly into the SVM classifier. The different score-spaces have been compared on a speaker-independent isolated letter task. The likelihood-ratio score-space out-performed the likelihood score-spaces and HMMs trained to maximise likelihood.

**Acknowledgements**

N. Smith would like to thank EPSRC; his CASE sponsor, the Speech Group at IBM U.K. Laboratories; and Thorsten Joachims, University of Dortmund, for $SVM^{light}$.

**References**

[1] V. Vapnik. *The Nature of Statistical Learning Theory.* Springer-Verlag, 1995.

[2] N. Smith, M. Gales, and M. Niranjan. Data-dependent kernels in SVM classification of speech patterns. Tech. Report CUED/F-INFENG/TR.387, Cambridge University Eng.Dept., April 2001.

[3] K. Tsuda et al. A New Discriminative Kernel from Probabilistic Models. In T.G. Dietterich, S. Becker and Z. Ghahramani, editors *Advances in Neural Information Processing Systems 14*, MIT Press, 2002.

[4] T. Jaakkola and D. Haussler. Exploiting Generative Models in Discriminative Classifiers. In M.S. Kearns, S.A. Solla, and D.A. Cohn, editors, *Advances in Neural Information Processing Systems 11*. MIT Press, 1999.

[5] N. Oliver, B. Schölkopf, and A. Smola. *Advances in Large-Margin Classifiers*, chapter Natural Regularization from Generative Models. MIT Press, 2000.

[6] S. Fine, J. Navrátil, and R. Gopinath. A hybrid GMM/SVM approach to speaker identification. In *Proceedings*, volume 1, International Conference on Acoustics, Speech, and Signal Processing, May 2001. Utah, USA.

[7] N. Smith and M. Gales. Using SVMs to classify variable length speech patterns. Tech. Report CUED/F-INFENG/TR.412, Cambridge University Eng.Dept., June 2001.

[8] M. Fanty and R. Cole. Spoken Letter Recognition. In R.P. Lippmann, J.E. Moody, and D.S. Touretzky, editors, *Neural Information Processing Systems 3*, pages 220–226. Morgan Kaufmann Publishers, 1991.

[9] T. Joachims. Making Large-Scale SVM Learning Practical. In B. Schölkopf, C. Burges, and A. Smola, editors, *Advances in Kernel Methods - Support Vector Learning*. MIT-Press, 1999.

[10] P.C. Loizou and A.S. Spanias. High-Performance Alphabet Recognition. *IEEE Transactions on Speech and Audio Processing*, 4(6):430–445, Nov. 1996.

## Footnotes

[1]For fuller details of the derivations see [2].

[2]Due to the sum to unity constraints, one of the weight parameters in each Gaussian mixture is discarded from the definition of the super-vector, as are the forward transitions in the left-to-right HMM with no skips.

[3]It is useful to note that a linear decision boundary, with zero bias, constructed in a single-dimensional likelihood-ratio score-space formed by the zeroth-order derivative operator would, under equal class priors, give the standard minimum Bayes error classifier.

[4]The error rate at 8.5% differed from that for the HMM baseline at 8.7% because of the non-zero bias for the SVMs.
